# Learning to be Bayesian without Supervision

**Martin Raphan**

Courant Inst. of Mathematical Sciences
New York University

raphan@cims.nyu.edu

**Eero P. Simoncelli**

Center for Neural Science, and
Courant Inst. of Mathematical Sciences
New York University

eero.simoncelli@nyu.edu

Bayesian estimators are defined in terms of the posterior distribution. Typically, this is written as the product of the likelihood function and a prior probability density, both of which are assumed to be known. But in many situations, the prior density is not known, and is difficult to learn from data since one does not have access to uncorrupted samples of the variable being estimated. We show that for a wide variety of observation models, the Bayes least squares (BLS) estimator may be formulated without explicit reference to the prior. Specifically, we derive a direct expression for the estimator, and a related expression for the mean squared estimation error, both in terms of the density of the observed measurements. Each of these prior-free formulations allows us to approximate the estimator given a sufficient amount of observed data. We use the first form to develop practical nonparametric approximations of BLS estimators for several different observation processes, and the second form to develop a parametric family of estimators for use in the additive Gaussian noise case. We examine the empirical performance of these estimators as a function of the amount of observed data.

## 1  Introduction

Bayesian methods are widely used throughout engineering for estimating quantities from corrupted measurements. Those that minimize the mean squared error (known as *Bayes least squares*, or BLS) are particularly widespread. These estimators are usually derived assuming explicit knowledge of the observation process (expressed as the conditional density of the observation given the quantity to be estimated), and the prior density over that quantity. Despite its appeal, this approach is often criticized for the reliance on knowledge of the prior distribution, since the true prior is usually not known, and in many cases one does not have data drawn from this distribution with which to approximate it. In this case, it must be learned from the same observed measurements that are available in the estimation problem. In general, learning the prior distribution from the observed data presents a difficult, if not impossible task, even when the observation process is known. In the commonly used "empirical Bayesian" approach [1], one assumes a parametric family of densities, whose parameters are obtained by fitting the data. This prior is then used to derive an estimator that may be applied to the data. If the true prior is not a member of the assumed parametric family, however, such estimators can perform quite poorly.

An estimator may also be obtained in a *supervised* setting, in which one is provided with many pairs containing a corrupted observation along with the true value of the quantity to be estimated. In this case, selecting an estimator is a classic regression problem: find a function that best maps the observations to the correct values, in a least squares sense. Given a large enough number of training samples, this function will approach the BLS estimate, and should perform well on new samples drawn from the same distribution as the training samples. In many real-world situations, however, one does not have access to such training data.

In this paper, we examine the BLS estimation problem in a setting that lies between the two cases described above. Specifically, we assume the observation process (but not the prior) is known, and

we assume *unsupervised* training data, consisting only of corrupted observations (without the correct values). We show that for many observation processes, the BLS estimator may be written directly in terms of the observation density. We also show a dual formulation, in which the BLS estimator may be obtained by minimizing an expression for the mean squared error that is written only in terms of the observation density. A few special cases of the first formulation appear in the empirical Bayes literature [2], and of the second formulation in another branch of the statistical literature concerned with improvement of estimators [3, 4, 5]. Our work serves to unify these prior-free methods within a linear algebraic framework, and to generalize them to a wider range of cases. We develop practical nonparametric approximations of estimators for several different observation processes, demonstrating empirically that they converge to the BLS estimator as the amount of observed data increases. We also develop a parametric family of estimators for use in the additive Gaussian case, and examine their empirical convergence properties. We expect such BLS estimators, constructed from corrupted observations without explicit knowledge of, assumptions about, or samples from the prior, to prove useful in a variety real-world estimation problems faced by machine or biological systems that must learn from examples.

## 2 Bayes least squares estimation

Suppose we make an observation, $Y$, that depends on a hidden variable $X$, where $X$ and $Y$ may be scalars or vectors. Given this observation, the BLS estimate of $X$ is simply the conditional expectation of the posterior density, $E\{X|Y = \mathbf{y}\}$. If the prior distribution on $X$ is $P_X$, and the likelihood function is $P_{Y|X}$ then this can be written using Bayes' rule as

$$
\begin{aligned}
E\{X|Y = \mathbf{y}\} &= \int \mathbf{x}\, P_{X|Y}(\mathbf{x}|\mathbf{y})\, d\mathbf{x} \\
&= \int \mathbf{x}\, P_{Y|X}(\mathbf{y}|\mathbf{x})\, P_X(\mathbf{x})\, d\mathbf{x} \Big/ P_Y(\mathbf{y}),
\end{aligned}
\tag{1}
$$

where the denominator is the distribution of the observed data:

$$
P_Y(\mathbf{y}) = \int P_{Y|X}(\mathbf{y}|\mathbf{x})\, P_X(\mathbf{x})\, d\mathbf{x}.
\tag{2}
$$

If we know $P_X$ and $P_{Y|X}$, we can calculate this explicitly.

Alternatively, if we do not know $P_X$ or $P_{Y|X}$, but are given independent identically distributed (i.i.d.) samples $(X_n, Y_n)$ drawn from the joint distribution of $(X, Y)$, then we can solve for the estimator $f(\mathbf{y}) = E\{X|Y = \mathbf{y}\}$ nonparametrically, or we could choose a parametric family of estimators $\{f_\theta\}$, and choose $\theta$ to minimize the empirical squared error:

$$
\hat{\theta} = \arg\min_\theta \frac{1}{N} \sum_{n=1}^{N} |f_\theta(Y_n) - X_n|^2.
$$

However, in many situations, one does not have access to $P_X$, or to samples drawn from $P_X$.

### 2.1 Prior-free reformulation of the BLS estimator

In many cases, the BLS estimate may be written without explicit reference to the prior distribution. We begin by noting that in Eq. (1), the prior appears only in the numerator

$$
N(\mathbf{y}) = \int P_{Y|X}(\mathbf{y}|\mathbf{x})\, \mathbf{x}\, P_X(\mathbf{x})\, d\mathbf{x}.
$$

This equation may be viewed as a composition of linear transformations of the function $P_X(\mathbf{x})$

$$
N(\mathbf{y}) = (\mathbf{A} \circ \mathbf{X})\{P_X\}(\mathbf{y}),
$$

where

$$
\mathbf{X}\{f\}(\mathbf{x}) = \mathbf{x}f(\mathbf{x}),
$$

and the operator $\mathbf{A}$ computes an inner product with the likelihood function

$$
\mathbf{A}\{f\}(\mathbf{y}) = \int P_{Y|X}(\mathbf{y}|\mathbf{x})\, f(\mathbf{x})\, d\mathbf{x}.
$$

Similarly, Eq. (2) may be viewed as the linear transformation $\mathbf{A}$ applied to $P_X(\mathbf{x})$. If the linear transformation $\mathbf{A}$ is 1-1, and we restrict $P_Y$ to lie in the range of $\mathbf{A}$, then we can then write the numerator as a linear transformation on $P_Y$ alone, without explicit reference to $P_X$:

$$
\begin{aligned}
N(\mathbf{y}) &= (\mathbf{A} \circ \mathbf{X} \circ \mathbf{A}^{-1})\{P_Y\}(\mathbf{y}) \\
&= \mathbf{L}\{P_Y\}(\mathbf{y}).
\end{aligned}
\tag{3}
$$

In the discrete case, $P_Y(\mathbf{y})$ and $N(\mathbf{y})$ are each vectors, $\mathbf{A}$ is a matrix containing $\mathbf{P}_{Y|X}$, $\mathbf{X}$ is a diagonal matrix containing values of $\mathbf{x}$, and $\circ$ is matrix multiplication.

This allows us to write the BLS estimator as

$$
E\{X|Y=\mathbf{y}\} = \frac{\mathbf{L}\{P_Y\}(\mathbf{y})}{P_Y(\mathbf{y})}.
\tag{4}
$$

Note that if we wished to calculate $E\{X^n|Y\}$, then Eq. (3) would be replaced by $(\mathbf{A} \circ \mathbf{X}^n \circ \mathbf{A}^{-1})\{P_Y\} = (\mathbf{A} \circ \mathbf{X} \circ \mathbf{A}^{-1})^n\{P_Y\} = \mathbf{L}^n\{P_Y\}$ . By linearity of the conditional expectation, we may extend this to any polynomial function (and thus to any function that can be approximated with a polynomial):

$$
E\left\{\sum_{k=-N}^{M} c_k X^k | Y = \mathbf{y}\right\} = \frac{\sum_{k=-N}^{M} c_k \mathbf{L}^k\{P_Y\}(\mathbf{y})}{P_Y(\mathbf{y})}.
$$

In the definition of the operator $\mathbf{L}$, $\mathbf{A}^{-1}$ effectively inverts the observation process, recovering $P_X$ from $P_Y$. In many situations, this operation will not be well-behaved. For example, in the case of additive Gaussian noise, $\mathbf{A}^{-1}$ is a deconvolution operation which is inherently unstable for high frequencies. The usefulness of Eq. (4) comes from the fact that in many cases, the composite operation $\mathbf{L}$ may be written explicitly, even when the inverse operation is poorly defined or unstable. In section 3, we develop examples of operators $\mathbf{L}$ for a variety of observation processes.

## 2.2 Prior-free reformulation of the mean squared error

In some cases, developing a stable nonparametric approximation of the ratio in Eq. (4) may be difficult. However, the linear operator formulation of the BLS estimator also leads to a dual expression for the mean squared error that does not depend explicitly on the prior, and this may be used to select an optimal estimator from a parametric family of estimators. Specifically, for any estimator $f_\theta(Y)$ parameterized by $\theta$, the mean squared error may be decomposed into two orthogonal terms:

$$
E\left\{|f_\theta(Y) - X|^2\right\} = E\left\{|f_\theta(Y) - E(X|Y)|^2\right\} + E\left\{|E(X|Y) - X|^2\right\}.
$$

The second term is the minimum possible MSE, obtained when using the optimal estimator. Since it does not depend on $f_\theta$, it is irrelevant for optimizing $\theta$. The first term may be expanded as

$$
E\left\{|f_\theta(Y) - E(X|Y)|^2\right\} = E\left\{|f_\theta(Y)|^2 - 2f_\theta(Y)E(X|Y)\right\} + E\left\{|E(X|Y)|^2\right\}.
$$

Again, the second expectation does not depend on $f_\theta$. Using the prior-free formulation of the previous section, the second component of the first expectation may be written as

$$
\begin{aligned}
E\{f_\theta(Y)E(X|Y)\} &= E\left\{f_\theta(Y)\frac{\mathbf{L}\{P_Y\}(Y)}{P_Y(Y)}\right\} \\
&= \int f_\theta(\mathbf{y})\frac{\mathbf{L}\{P_Y\}(\mathbf{y})}{P_Y(\mathbf{y})}P_Y(\mathbf{y})d\mathbf{y} \\
&= \int f_\theta(\mathbf{y})\,\mathbf{L}\{P_Y\}(\mathbf{y})d\mathbf{y} \\
&= \int \mathbf{L}^*\{f_\theta\}(\mathbf{y})P_Y(\mathbf{y})d\mathbf{y} \\
&= E\{\mathbf{L}^*\{f_\theta\}(Y)\},
\end{aligned}
$$

where $\mathbf{L}^*$ is the dual operator of $\mathbf{L}$ (in the discrete case, $\mathbf{L}^*$ is the matrix transpose of $\mathbf{L}$). Combining all of the above, we have:

$$\arg \min_\theta E \left\{ |f_\theta(Y) - X|^2 \right\} = \arg \min_\theta E \left\{ |f_\theta(Y)|^2 - 2\mathbf{L}^*\{f_\theta\}(Y) \right\}. \tag{5}$$

where the expectation on the right is over the observation variable, $Y$. In practice, we can solve for an optimal $\theta$ by minimizing the sample mean of this quantity:

$$\hat{\theta} = \arg \min_\theta \frac{1}{N} \sum_{n=1}^N \left\{ |f_\theta(Y_n)|^2 - 2\mathbf{L}^*\{f_\theta\}(Y_n) \right\}. \tag{6}$$

where $\{Y_n\}$ is a set of observed data. Again this does not require any knowledge of, or samples drawn from, the prior $P_X$.

## 3   Example estimators

In general, it can be difficult to obtain the operator $\mathbf{L}$ directly from the definition in Eq. (3), because inversion of the operator $\mathbf{A}$ could be unstable or undefined. Instead, a solution may often be obtained by noting that the definition implies that

$$\mathbf{L} \circ \mathbf{A} = \mathbf{A} \circ \mathbf{X},$$

or, equivalently

$$\mathbf{L}\{P_{Y|X}(\mathbf{y}|\mathbf{x})\} = \mathbf{x} P_{Y|X}(\mathbf{y}|\mathbf{x}).$$

This is an eigenfunction equation: for each value of $\mathbf{x}$, the conditional density $P_{Y|X}(\mathbf{y}|\mathbf{x})$ must be an eigenfunction (eigenvector, for discrete variables) of eoperator $\mathbf{L}$, with associated eigenvalue $\mathbf{x}$.

Consider a standard example, in which the variable of interest is corrupted by independent additive noise: $Y = X + W$. The conditional density is

$$P_{Y|X}(\mathbf{y}|\mathbf{x}) = P_W(\mathbf{y} - \mathbf{x}).$$

We wish to find an operator which when applied to this conditional density (viewed as a function of $\mathbf{y}$) will give

$$\mathbf{L}\{P_W(\mathbf{y} - \mathbf{x})\} = \mathbf{x}\, P_W(\mathbf{y} - \mathbf{x}) \tag{7}$$

for all $\mathbf{x}$. Subtracting $\mathbf{y}\, P_W(\mathbf{y} - \mathbf{x})$ from both sides gives

$$\mathbf{M}\{P_W(\mathbf{y} - \mathbf{x})\} = -(\mathbf{y} - \mathbf{x})\, P_W(\mathbf{y} - \mathbf{x}). \tag{8}$$

where

$$\mathbf{M}\{f\}(\mathbf{y}) = \mathbf{L}\{f\}(\mathbf{y}) - \mathbf{y}\, f(\mathbf{y})$$

is a linear shift-invariant operator (acting in $\mathbf{y}$).

Taking Fourier transforms and using the convolution and differentiation properties gives:

$$\begin{aligned}
\widehat{\mathbf{M}}(\omega)\widehat{P_W}(\omega) &= -(\widehat{\mathbf{y} P_W})(\omega) \\
&= -i\nabla_\omega \widehat{P_W}(\omega), \tag{9}
\end{aligned}$$

so that

$$\begin{aligned}
\widehat{\mathbf{M}}(\omega) &= -i\frac{\nabla_\omega \widehat{P_W}(\omega)}{\widehat{P_W}(\omega)} \\
&= -i\nabla_\omega \ln\left(\widehat{P_W}(\omega)\right). \tag{10}
\end{aligned}$$

This gives us the linear operator

$$\mathbf{L}\{f\}(\mathbf{y}) = \mathbf{y}\, f(\mathbf{y}) - \mathcal{F}^{-1}\left\{ i\nabla_\omega \ln\left(\widehat{P_W}(\omega)\right) \widehat{f}(\omega) \right\}(\mathbf{y}), \tag{11}$$

where $\mathcal{F}^{-1}$ denotes the inverse Fourier transform. Note that throughout this discussion $X$ and $W$ played symmetric roles. Thus, in cases with known prior density and unknown additive noise density, one can formulate the estimator entirely in terms of the prior.

Our prior-free estimator methodology is quite general, and can often be applied to more complicated observation processes. In order to give some sense of the diversity of forms that can arise, Table 1 provides additional examples. References for the specific cases that we have found in the statistics literature are provided in table.

| Obs. process | Obs. density: $P_{Y|X}(\mathbf{y}|\mathbf{x})$ | Numerator: $N(\mathbf{y}) = \mathbf{L}\{P_Y\}(\mathbf{y})$ |
|---|---|---|
| Discrete | $\mathbf{A}$ | $(\mathbf{A} \circ X \circ \mathbf{A}^{-1})P_Y(\mathbf{y})$ |
| Gen. add. | $P_W(y - x)$ | $\mathbf{y}P_Y - \mathcal{F}^{-1}\left\{ i\nabla_\omega \ln\left(\widehat{P_W}(\omega)\right)\widehat{P_Y}(\omega)\right\}$ |
| Add. Gaussian [6]/[4]* | $\dfrac{\exp\left\{\frac{-1}{2}(\mathbf{y}-\mathbf{x}-\mu)^T\Lambda^{-1}(\mathbf{y}-\mathbf{x}-\mu)\right\}}{\sqrt{\lvert 2\pi\Lambda\rvert}}$ | $(\mathbf{y}-\mu)P_Y(\mathbf{y}) + \Lambda\nabla_\mathbf{y}P_Y(\mathbf{y})$ |
| Add. Poisson | $\sum \frac{\lambda^k e^{-\lambda}}{k!}\delta(y-x-ks)$ | $yP_Y(y) - \lambda s P_Y(y-s)$ |
| Add. Laplacian | $\frac{1}{2\alpha}e^{-\lvert(y-x)/\alpha\rvert}$ | $yP_Y(y) + 2\alpha^2\{P_W' \star P_Y\}(y)$ |
| Add. Cauchy | $\frac{1}{\pi}\left(\frac{\alpha}{(\alpha(y-x))^2+1}\right)$ | $yP_Y(y) - \{\frac{1}{2\pi\alpha y} \star P_Y\}(y)$ |
| Add. uniform | $\begin{cases} \frac{1}{2a}, & \lvert y-x\rvert \le a \\ 0, & \lvert y-x\rvert > a \end{cases}$ | $yP_Y(y) + a\sum \mathrm{sgn}(k)P_Y(y-ak)$ $-\frac{1}{2}\int P_Y(\tilde y)\mathrm{sgn}(y-\tilde y)d\tilde y$ |
| Add. random # of components | $P_W(y-x)$, where: $W \sim \sum_{k=0}^K W_k,$ $W_k$ i.i.d. $(P_c),$ $K \sim \mathrm{Poiss}(\lambda)$ | $yP_Y(y) - \lambda\{(yP_c) \star P_Y\}(y)$ |
| Disc. exp. [2]/[5]* | $h(x)g(n)x^n$ | $\frac{g(n)}{g(n+1)}P_Y(n+1)$ |
| Disc. inv. exp. [5]* | $h(x)g(n)x^{-n}$ | $\frac{g(n)}{g(n-1)}P_Y(n-1)$ |
| Cnt. exp. [2]/[3]* | $h(x)g(y)e^{T(y)x}$ | $\frac{g(y)}{T'(y)}\frac{d}{dy}\{\frac{P_Y(y)}{g(y)}\}$ |
| Cnt. inv. exp. [3]* | $h(x)g(y)e^{T(y)/x}$ | $g(y)\int_{-\infty}^y \frac{T'(\tilde y)}{g(\tilde y)}P_Y(\tilde y)d\tilde y$ |
| Poisson [7]/[5]* | $\frac{x^n e^{-x}}{n!}$ | $(n+1)P_Y(n+1)$ |
| Gauss. scale mixture | $\frac{1}{\sqrt{2\pi x}}e^{-\frac{y^2}{2x}}$ | $-E_Y\{Y; Y < y\}$ |
| Lapl. scale mixture | $\frac{1}{x}e^{-\frac{y}{x}}; x, y > 0$ | $P_Y\{Y > y\}$ |

Table 1: Prior-free estimation formulas. Functions written with hats or in terms of $\omega$ represent multiplication in the Fourier Domain. $n$ replaces $y$ for discrete distributions. Bracketed numbers are references for operators $\mathbf{L}$, with * denoting references for the parametric (dual) operator, $\mathbf{L}^*$.

# 4 Simulations

## 4.1 Non-parametric examples

Since each of the prior-free estimators discussed above relies on approximating values from the observed data, the behavior of such estimators should approach the BLS estimator as the number of data samples grows. In Fig. 1, we examine the behavior of three non-parametric prior-free estimators based on Eq. (4). The first case corresponds to data drawn independently from a binary source, which are observed through a process in which bits are switched with probability $\frac{1}{4}$. The estimator does not know the binary distribution of the source (which was a "fair coin" for our simulation), but does know the bit-switching probability. For this estimator we approximate $P_Y$ using a simple histogram, and then use the matrix version of the linear operator in Eq. (3). We characterize the behavior of this estimator as a function of the number of data points, $N$, by running many Monte Carlo simulations for each $N$ and indicating the mean improvement in MSE (compared with the ML estimator, which is the identity function), the mean improvement using the conventional BLS estimation function, $E\{X|Y=\mathbf{y}\}$ assuming the prior density is known, and the standard deviations of the improvements taken over our simulations.

Figure 1**b** shows similar results for additive Gaussian noise, with SNR replacing MSE. Signal density is a generalized Gaussian with exponent $0.5$. In this case, we compute Eq. (4) using a more

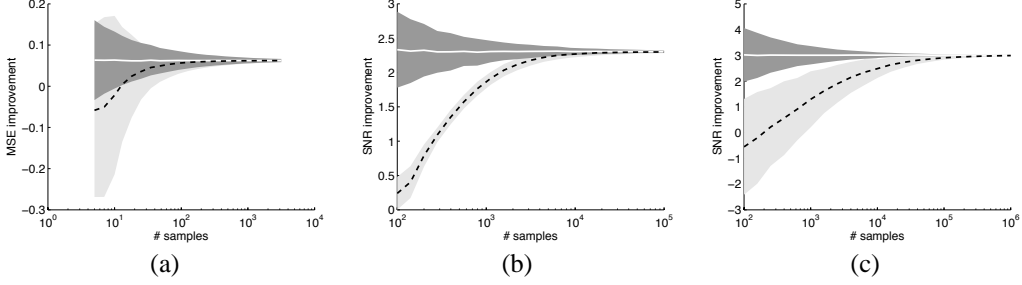

Fig. 1: Empirical convergence of prior-free estimator to optimal BLS solution, as a function number of observed samples of $Y$. For each number of observations, each estimator is simulated many times. Black dashed lines show the improvement of the prior-free estimator, averaged over simulations, relative to the ML estimator. White line shows the mean improvement using the conventional BLS solution, $E\{X|Y = \mathbf{y}\}$, assuming the prior density is known. Gray regions denote $\pm$ one standard deviation. (**a**) Binary noise (10,000 simulations for each number of observations); (**b**) additive Gaussian noise (1,000 simulations); (**c**) Poisson noise (1,000 simulations).

sophisticated approximation method, as described in [8]. We fit a local exponential model similar to that used in [9] to the data in bins, with binwidth adaptively selected so that the product of the number of points in the bin and the squared binwidth is constant. This binwidth selection procedure, analogous to adaptive binning procedures developed in the density estimation literature [10], provides a reasonable tradeoff between bias and variance, and converges to the correct answer for any well-behaved density [8]. Note that in this case, convergence is substantially slower than for the binary case, as might be expected given that we are dealing with a continuous density rather than a single scalar probability. But the variance of the estimates is very low.

Figure 1**c** shows the case of estimating a randomly varying rate parameter that governs an inhomogeneous Poisson process. The prior on the rate (unknown to the estimator) is exponential. The observed values $Y$ are the (integer) values drawn from the Poisson process. In this case the histogram of observed data was used to obtain a naive approximation of $P_Y(n)$. It should be noted that improved performance for this estimator is expected if we were to use a more sophisticated approximation of the ratio of densities.

## 4.2 Parametric examples

In this section we discuss the empirical behavior of the parametric approach applied to the additive Gaussian case. From the derivation in section 3, and restricting to the scalar case, we have

$$\mathbf{L}^* = y - \sigma^2 \frac{d}{dy}.$$

In this particular case,, it is easier to represent the estimator as

$$f(y) = y + g(y).$$

Substituting into Eq. (5) gives

$$E\{|f(Y) - X|^2\} = E\{|g(Y)|^2 + \sigma^2 g'(Y)\} + \text{const},$$

where the constant does not depend on $g$. Therefore, if we have a parametric family $\{g_\theta\}$ of such $g$, and are given data $\{Y_n\}$ we can try and minimize

$$\frac{1}{N} \sum_{n=1}^{N} \{|g_\theta(Y_n)|^2 + \sigma^2 g'_\theta(Y_n)\}. \tag{12}$$

This expression, known as Stein's unbiased risk estimator (SURE) [4], favors estimators $g_\theta$ that have small amplitude, and highly negative derivatives at the data values. This is intuitively sensible: the resulting estimators will "shrink" the data toward regions of high probability.

Recently, an expression similar to Eq. (12) was used as a criterion for density estimation in cases where the normalizing constant, or partition function, is difficult to obtain [11]. The prior-free

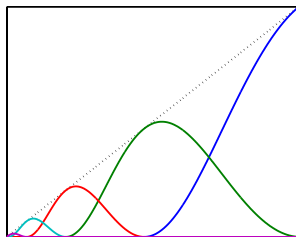

Fig. 2: Example bump functions, used for linear parameterization of estimators in Figs. 3(a) and 3(b).

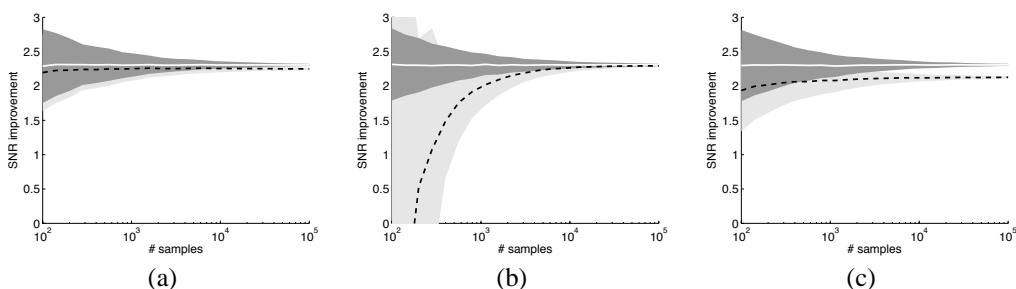

(a)                          (b)                          (c)

Fig. 3: Empirical convergence of parametric prior-free method to optimal BLS solution, as a function number of data observations, for three different parameterized estimators. (**a**)3 bump; (**b**)15 bumps; (**c**) Soft thresholding. All cases use a generalized Gaussian prior (exponent 0.5), and assume additive Gaussian noise.

approach we are discussing provides an interpretation for this procedure: the optimal density is the one which, when converted into an estimator using the formula in Table 1 for the additive Gaussian case, gives the best MSE. This may be extended to any of the linear operators in Table 1.

As an example, we parametrize $g$ as a linear combination of nonlinear "bump" functions

$$g_\theta(y) = \sum_k \theta_k g_k(y) \tag{13}$$

where the functions $g_k$ are of the form

$$g_k(y) = y \; \cos^2\left(\frac{1}{\alpha}\mathrm{sgn}(y)\log_2(|y|/\sigma + 1) - \frac{k\pi}{2}\right),$$

as illustrated in Fig. 2. Recently, linear parameterizations have been used in conjunction with Eq. (12) for image denoising in the wavelet domain [12].

We can substitute Eq. (13) into Eq. (12) and pick coefficients $\{\theta_k\}$ to minimize this criteria, which is a quadratic function of the coefficients. For our simulations, we used a generalized Gaussian prior, with exponent 0.5. Figure 3 shows the empirical behavior of these "SURE-bump" estimators when using three bumps ( Fig. 3**a**) and fifteen bumps (Fig. 3**b**), illustrating the bias-variance trade-off inherent in the fixed parameterization. Three bumps behaves fairly well, though the asymptotic behavior for large amounts of data is biased and thus falls short of ideal. Fifteen bumps asymptotes correctly but has very large variance for small amounts of data (overfitting). For comparison purposes, we have included the behavior of SURE thresholding [13], in which Eq. (4.2) is used to choose an optimal threshold, $\theta$, for the function

$$f_\theta(y) = \mathrm{sgn}(y)(|y| - \theta)^+.$$

As can be seen, SURE thresholding shows significant asymptotic bias although the variance behavior is nearly ideal.

## 5   Discussion

We have reformulated the Bayes least squares estimation problem for a setting in which one knows the observation process, and has access to many observations. We do not assume the prior density

is known, nor do we assume access to samples from the prior. Our formulation thus acts as a bridge between a conventional Bayesian setting in which one derives the optimal estimator from known prior and likelihood functions, and a data-oriented regression setting in which one learns the optimal estimator from samples of the prior paired with corrupted observations of those samples. In many cases, the prior-free estimator can be written explicitly, and we have shown a number of examples to illustrate the diversity of estimators that can arise under different observation processes. For three simple cases, we developed implementations and demonstrated that these converge to optimal BLS estimators as the amount of data grows. We also have derived a prior-free formulation of the MSE, which allows selection of an estimator from a parametric family. We have shown simulations for a linear family of estimators in the additive Gaussian case.

These simulations serve to demonstrate the potential of this approach, which holds particular appeal for real-world systems (machine or biological) that must learn the priors from environmental observations. Both methods can be enhanced by using data-adaptive parameterizations or fitting procedures in order to properly trade off bias and variance (see, for example [8]). It is of particular interest to develop incremental implementations, which would update the estimator based on incoming observations. This would further enhance the applicability of this approach for systems that must learn to do optimal estimation from corrupted observations.

### Acknowledgments

This work was partially funded by the Howard Hughes Medical Institute, and by New York University through a McCracken Fellowship to MR.

## References

[1] G. Casella, "An introduction to empirical Bayes data analysis," *Amer. Statist.*, vol. 39, pp. 83–87, 1985.

[2] J. S. Maritz and T. Lwin, *Empirical Bayes Methods*. Chapman & Hall, 2nd ed., 1989.

[3] J. Berger, "Improving on inadmissible estimators in continuous exponential families with applications to simultaneous estimation of gamma scale parameters," *The Annals of Staistics*, vol. 8, pp. 545–571, 1980.

[4] C. M. Stein, "Estimation of the mean of a multivariate normal distribution," *Annals of Statistics*, vol. 9, pp. 1135–1151, November 1981.

[5] J. T. Hwang, "Improving upon standard estimators in discrete exponential families with applications to poisson and negative binomial cases," *The Annals of Staistics*, vol. 10, pp. 857–867, 1982.

[6] K. Miyasawa, "An empirical bayes estimator of the mean of a normal population," *Bull. Inst. Internat. Statist.*, vol. 38, pp. 181–188, 1956.

[7] H. Robbins, "An empirical bayes approach to statistics," *Proc. Third Berkley Symposium on Mathematcal Statistics*, vol. 1, pp. 157–163, 1956.

[8] M. Raphan and E. P. Simoncelli, "Empirical Bayes least squares estimation without an explicit prior." NYU Courant Inst. Tech. Report, 2007.

[9] C. R. Loader, "Local likelihood density estimation," *Annals of Statistics*, vol. 24, no. 4, pp. 1602–1618, 1996.

[10] D. W. Scott, *Multivariate Density Estimation: Theory, Practice, and Visualization*. John Wiley, 1992.

[11] A. Hyvarinen, "Estimation of non-normalized statistical models by score matching," *Journal of Machine Learning Research*, vol. 6, pp. 695–709, 2005.

[12] F. Luisier, T. Blu, and M. Unser, "SURE-based wavelet thresholding integrating inter-scale dependencies," in *Proc IEEE Int'l Conf on Image Proc*, (Atlanta GA, USA), pp. 1457–1460, October 2006.

[13] D. Donoho and I. Johnstone, "Adapting to unknown smoothness via wavelet shrinkage," *J American Stat Assoc*, vol. 90, December 1995.
